# Kernel-ARMA for Hand Tracking and Brain-Machine Interfacing During 3D Motor Control

**Lavi Shpigelman**[1] **, Hagai Lalazar** [2] **and Eilon Vaadia** [3]
Interdisciplinary Center for Neural Computation
The Hebrew University of Jerusalem, Israel
[1]shpigi@gmail.com, [2]hagai@alice.nc.huji.ac.il,
[3]eilonv@ekmd.huji.ac.il

## Abstract

Using machine learning algorithms to decode intended behavior from neural activity serves a dual purpose. First, these tools allow patients to interact with their environment through a Brain-Machine Interface (BMI). Second, analyzing the characteristics of such methods can reveal the relative significance of various features of neural activity, task stimuli, and behavior. In this study we adapted, implemented and tested a machine learning method called Kernel Auto-Regressive Moving Average (KARMA), for the task of inferring movements from neural activity in primary motor cortex. Our version of this algorithm is used in an online learning setting and is updated after a sequence of inferred movements is completed. We first used it to track real hand movements executed by a monkey in a standard 3D reaching task. We then applied it in a closed-loop BMI setting to infer *intended* movement, while the monkey's arms were comfortably restrained, thus performing the task using the BMI alone. KARMA is a recurrent method that learns a nonlinear model of output dynamics. It uses similarity functions (termed kernels) to compare between inputs. These kernels can be structured to incorporate domain knowledge into the method. We compare KARMA to various state-of-the-art methods by evaluating tracking performance and present results from the KARMA based BMI experiments.

## 1   Introduction

Performing a behavioral action such as picking up a sandwich and bringing it to one's mouth is a motor control task achieved easily every day by millions of people. This simple action, however, is impossible for many patients with motor deficits. In the future, patients with enough cortical activity remaining may benefit from Brain Machine Interfaces that will restore motor control with agility, precision, and the degrees of freedom comparable to natural movements. Such high quality BMI's are not yet available. The BMI framework involves recording neural activity, typically using chronically implanted electrodes, which is fed in real-time to a decoding algorithm. Such algorithms attempt to infer the subject's intended behavior. The algorithm's predictions can be used to artificially control an end-effector: a cursor on a screen, a prosthetic arm, a wheelchair, or the subject's own limbs by stimulation of their muscles. This study focuses on the algorithmic component.

Motor control is a dynamic process involving many feedback loops, relevant time frames, and constraints of the body and neural processing. Neural activity in primary motor cortex (MI) is part of this process. An early approach at decoding movement from MI activity for BMI (see [1]) was rather simplistic. Instantaneous velocity of the cursor, across a set of movements, was linearly regressed against neuronal spike rates. This algorithm (known as the Population Vector Algorithm) is equivalent to modelling each neuron as a consine function of movement velocity. This method is still used today for BMI's [2], and has become the standard model in many studies of encoding and learning in MI. Our understanding of motor cortex has progressed, and many other factors have been shown to

correlate with neuronal activity, but are typically overlooked in modeling. For example, MI activity has been shown to encode arm posture [3], the dynamic aspects of the movement (such as current acceleration, or interaction forces) and the interactions between neurons and their dynamics [4].

State-of-the-art movement decoding methods typically involve improved modeling of behavior, neural activity, and the relations between them. For example, Kalman filtering (see [5]) has been used to model the system state as being comprised of current hand position, velocity and acceleration. Thus, the hand movement is assumed to have roughly constant acceleration (with added Gaussian noise and, consequently, minimal jerk) and the neural activity is assumed to be a linear function of the hand state (with added Gaussian noise). Particle filtering, which relaxes some of the linearity and Gaussian assumptions, has also been applied in an offline setting (see [6]). Support Vector Regression (SVR) from neural activity to current hand velocity (see [7]) has the advantage of allowing for extraction of nonlinear information from neuronal interactions, but is missing a movement model. One of our previous studies ([8]) combines a linear movement model (as in Kalman filtering) with SVR-based nonlinear regression from neural activity.

KARMA (see [9] for one of its first appearances, or [10] for a more recent one) is a kernelized version of the ARMA method [11]. It performs ARMA in a kernel-induced feature space (for a comprehensive explanation of this *kernel-trick,* see [12]). It estimates the next system state as a function of both the time window of previous state estimates (the Auto-Regressive part) and the time window of previous observations (the Moving-Average part). In our application, we extend its formulation to the Multi-Input Multi-Output (MIMO) case, allowing for better modeling of the system state. We apply it in an online learning paradigm, and by limiting the number of support vectors turn it into an adaptive method. This allows for real-time inference, as is necessary for BMI. In section 2 we explain the lab setup, the problem setting, and introduce our notation. In section 3 we describe KARMA, and our online and adaptive version of it, in detail. We explain the available modeling options and how they can be used to either improve performance or to test ideas regarding motor control and neural encoding. Section 4 describes KARMA's performance in tracking hand movements and compares it with other state-of-the-art methods. Section 5 presents results from our BMI experiment using KARMA and, finally, we summarize in section 6

## 2   Lab setup and problem setting

In our experiments, a monkey performed a visuomotor control task that involved moving a cursor on a screen from target to target in 3D virtual space. Neuronal activity from a population of single and multi-units was recorded with a chronically implanted array of 96 electrodes (Cyberkinetics, Inc.) in MI, and used to calculate spike rates (spike counts in 50ms time bins smoothed by a causal filter). In *hand-control* (open-loop) mode, the monkey used its hand (continuously tracked by an optical motion capture system; Phoenix Tech., Inc.) to move the cursor. Data collected from these sessions is used here to assess algorithm performance, by using the real arm movements as the target trajectories. In hand-control, behavior is segmented into sequences of continuous recordings, separated by time periods during which the monkey's hand is not in view of the tracking device (e.g. when the monkey stops to scratch itself). Each sequence is made up of target-to-target reaching trials (some successful and some not). The targets appeared randomly in the 27 corners, centers of faces, and middle of a virtual cube whose side was 6cm. The target radii were 2.4cm. A successful trial consisted of one reaching movement that started at rest in one target and ended at rest in the next target (with required target hold periods during which the cursor must not leave the target). The next target appears at some point during the hold period of the previous target. Food reward is provided through a tube after each success. In case of failure, the cursor disappears for 1-2 seconds (failure period). During this time the monkey's hand is still tracked. In the *BMI* (closed-loop) setting, the monkey's hands were comfortably restrained and the KARMA algorithm's inference was used to move the cursor. Trial failures occured if the reaching movement took longer than 6 seconds or if the cursor was not kept inside the target during a 0.25s hold period. During trial-failure periods the inference was stopped and at the next trial the cursor reappeared where it left off. The trial-failure period was also used to pass the latest recorded trial sequence to the model-learning process, and to replace the working model with an updated one, if available.

In this text, $\mathbf{X}$ (Capital, bold) is a matrix, $\mathbf{x}$ (bold) is a vector ( so is $\mathbf{x}^i$, $\mathbf{x}_t$ or $\mathbf{x}_t^i$) and $x$ is a scalar. $(\mathbf{x})^T$ signifies transposition. We will use $\mathbf{x}_t^i \in \mathbb{R}^q$ to designate the neural activity (of $q$ cortical units) at time bin $t$ in behavior sequence $i$, which we refer to as *observations*. Given a window size,

$s$, $\mathbf{x}^i_{t-s+1:t} = \left( \left( \mathbf{x}^i_{t-s+1} \right)^T , \ldots , \left( \mathbf{x}^i_t \right)^T \right)^T \in I\!\!R^{sq}$ is an $sq$-long vector comprising a concatenated window of observations ending at time $t$, of trajectory $i$. $\mathbf{x}^i$ will be short-hand notation meaning $\mathbf{x}^i_{1:t_{f_i}}$ where $t_{f_i}$ is the number of steps in the whole $i$th trajectory. Similarly, $\mathbf{y}^i_t \in I\!\!R^d$ are used to designate cursor position ($d = 3$). We refer to $\mathbf{y}^i_t$ as the *state* trajectory. Given a window size, $r$ ,$\mathbf{y}^i_{t-r:t-1} \in I\!\!R^{rd}$ is a concatenated vector of states. *Estimated* states are differentiated from *true* states (or *desired* states, as will be explained later) by addition of a hat: $\hat{\mathbf{y}}^i_t$. Furthermore (given $s$ and $r$) we will use $\hat{\mathbf{v}}^i_t = \left( \left( \hat{\mathbf{y}}^i_{t-r:t-1} \right)^T \left( \mathbf{x}^i_{t-s+1:t} \right)^T \right)^T \in I\!\!R^{rd+sq}$ to concatenate *windows* of *estimated* states and of neural observations and $\mathbf{v}^i_t$ to concatenate *true* (rather than estimated) state values.

In the hand-control setting, we are given a (fully observed) data-set of neural activities and state trajectories: $\left\{ \mathbf{x}^i, \mathbf{y}^i \right\}^n_{i=1}$. Our goal is to learn to reconstruct the state trajectories from the neural activities. We adhere to the online learning paradigm in which at each step, $i$, of the process we are given one observation sequence, $\mathbf{x}^i$, predict $\hat{\mathbf{y}}^i$, then receive the *true* $\mathbf{y}^i$ and update the model. This allows the model to adapt to changes in the input-output relation that occurs over time.

In BMI mode, since hand movements were not performed, we do not know the *correct* cursor movement. Instead, during learning we use the cursor movement generated in the BMI and the positions of the targets that the monkey was instructed to reach to guess a *desired* cursor trajectory which is used to replace the missing true trajectory as feedback. The illustration on the right shows the BMI setup from an algorithmic point of view.

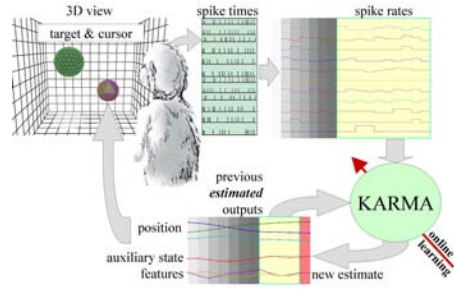

## 3   KARMA, modeling options, and online learning

As stated earlier, KARMA is a kernelized ARMA. In ARMA: $\mathbf{y}^i_t = \sum^r_{k=1} \mathbf{A}_k \mathbf{y}^i_{t-k} + \sum^s_{l=1} \mathbf{B}_l \, \mathbf{x}^i_{t-l+1} + \mathbf{e}^i_t$, where $\{ \mathbf{A}_k \}^r_{k=1}$ and $\{ \mathbf{B}_l \}^s_{l=1}$ are the respective Auto-Regressive (AR) and Moving Average (MA) parameters and $\mathbf{e}^i_t$ are residual error terms. Given these model parameters and initial state values, the rest of the state trajectory can be estimated from the observations by recursive application, replacing true state values with the estimated ones. Thus, ARMA inference is essentially application of a linear (MIMO) IIR filter. Defining $\mathbf{W} = [\mathbf{A}_r, \ldots, \mathbf{A}_1, \mathbf{B}_s, \ldots, \mathbf{B}_1]$, the next state estimate is simply $\hat{\mathbf{y}}^i_t = \mathbf{W} \hat{\mathbf{v}}^i_t$ (see notation section).

Kernelizing ARMA involves application of the kernel trick. A kernel function $k \left( \mathbf{v}_1, \mathbf{v}_2 \right)$ : $I\!\!R^{rd+sq} \times I\!\!R^{rd+sq} \to I\!\!R$ is introduced, which, conceptually, can be viewed as a dot product of feature vectors: $k \left( \mathbf{v}_1, \mathbf{v}_2 \right) = \boldsymbol{\phi}^T \left( \mathbf{v}_1 \right) \boldsymbol{\phi} \left( \mathbf{v}_2 \right)$ where the features are possibly complicated functions of both states and (neural) observations. Inference takes the form $\hat{\mathbf{y}}^i_t = \sum_{j,\tau} \boldsymbol{\alpha}^j_\tau k \left( \hat{\mathbf{v}}^i_t, \mathbf{v}^j_\tau \right)$ where $\boldsymbol{\alpha}^j_\tau \in I\!\!R^d$ are learned weight vectors and $\mathbf{v}^j_\tau$ are examples from a training set, known as the support set. Conceptually, KARMA inference can be viewed as $\hat{\mathbf{y}}^i_t = \mathbf{W}_\phi \boldsymbol{\phi} \left( \hat{\mathbf{v}}^i_t \right)$ where, as compared with ARMA, $\hat{\mathbf{v}}^i_t$ is replaced by its feature vector, $\mathbf{W}$ is replaced by $\mathbf{W}_\phi = \sum_{j,\tau} \boldsymbol{\alpha}^j_\tau \boldsymbol{\phi}^T (\mathbf{v}^j_\tau)$ and each recursive step of KARMA is linear regression in the feature space of observations + states. The weights, $\left\{ \boldsymbol{\alpha}^j_\tau \right\}$ are learned so as to solve the following optimization problem (presented in its primal form): $\arg \min_{\mathbf{W}_\phi} \| \mathbf{W}_\phi \|^2 + c \sum_{i,t,k} \left| \left( \mathbf{y}^i_t \right)_k - \left( \mathbf{W}_\phi \boldsymbol{\phi} \left( \mathbf{v}^i_t \right) \right)_k \right|_\epsilon$, where $\| \mathbf{w} \|^2 = \sum_{a,b} \left( \mathbf{W} \right)^2_{ab}$ is the Frobenius matrix norm, the sum in the second term is over all trials, times and state dimensions of the examples in the training set, $|v|_\epsilon = \max \{ 0, |v| - \epsilon \}$ is the $\epsilon$-insensitive absolute error and $c$ is a constant that determines the relative trade-off between the first (regularization) term and the second (error) term. Note that during learning, the states are estimated using the *true / desired* previous state values as input instead of the estimated ones (contrary to what is done during inference). "Luckily", this optimization problem reduces to standard SVR where $\mathbf{x}^i_t$ is replaced with $\mathbf{v}^i_t$. This replacement can be done as a preprocessing step in learning and a standard SVR solver can then be used to find the weights. Inference would require plugging in the previously estimated state values as part of the inputs between iterative calls to SVR inference.

Application of KARMA to a specific domain entails setting of some key hyper-parameters that may have drastic effects on performance. The relatively simple ones are the window sizes ($r$ and

$s$) and trade-off parameter $c$. Beyond the necessary selection of the cursor trajectories as the states, augmenting state dimensions (whose values are known at training and inferred during model testing) can be added in order to make the model use them as explicit features. This idea was tried in our hand tracking experiments using features such as absolute velocity and current trial state (reaching target, holding at target and trial-failure time). But since results did not improve significantly, we discontinued this line of research. The kernel function and its parameters must also be chosen. Note that the kernel in this algorithm is over structured data, which opens the door to a plethora of choices. Depending on one's view this can be seen as an optimization curse or as a modeling blessing. It obviously complicates the search for effective solutions but it allows to introduce domain knowledge (or assumptions) into the problem. It can also be used as a heuristic for *testing* the relative contribution of the assumptions behind the modeling choices. For example, by choosing $r = 0$ the algorithm reduces to SVR and the state model (and its dynamics) are ignored. By selecting a kernel which is a linear sum of two kernels, one for states and one for observations, the user assumes that states and observations have no "synergy" (i.e. each series can be read without taking the other into account). This is because summing of kernels is equivalent to calculating the features on their inputs separately and then concatenating the feature vectors. Selecting linear kernels reduces KARMA to ARMA (using its regularized loss function).

In online learning, one may change the learned model between consecutive inferences of (whole) time series. At learning step $k$, all of $\left\{ \mathbf{x}^i, \mathbf{y}^i \right\}_{i=1}^{k}$ are available for training. A naive solution would be to re-learn the model at every step, however, this would not be the best solution if one believes that the source of the input-output relation is changing (for example, in our BMI, cortical units may change their response properties, or units may appear or disappear during a recording session). Also, it may not be feasible to store all the sequences, or learning may take too long (opening up a delay between data acquisition until a new model is ready). If the resulting model has too many support vectors, too much time is required for each inference step (which is less than 50ms in our BMI setup). We deal with all the issues above by limiting the number of examples ($\mathbf{v}_t^i$) that are kept in memory (to 5000 in hand-control tracking and 3000 for real-time use in the BMI). At each online learning iteration, the latest data is added to the pool of examples one example at a time, and if the limit has been reached another example is selected at random (uniformly over the data-set) and thrown out. This scheme gives more mass to recent observations while allowing for a long tail of older observations. For a 3000 sized database and examples coming in at the rate of one per 50ms, the cache is filled after the first 150 seconds. Afterwards, the half life (the time required for an example to have a 50% chance of being thrown out) of an example is approximately 104 seconds, or conversely, at each point, approx. 63% of the examples in the database are from the last 150 seconds and the rest are earlier ones. This scheme keeps the inference time to a constant and seems reasonable in terms of rate of adaptation. We chose 5000 for the tracking (hand-control) experiments since in those experiments there is no real-time inference constraint and the performance improves a bit (suggesting that the 3000 size is not optimal in terms of inference quality). The similarity between consecutive examples is rather high as they share large parts of their time windows (when ARMA parameters $r$ or $s$ are large). Throwing away examples at random has a desired effect of lessening the dependency between remaining examples.

## 4   Open-loop hand tracking testing

To test various parametrization of KARMA and to compare its performance to other methods we used data from 11 hand-control recording days. These sessions vary in length from between 80 to 120 minutes of relatively continuous performance on the part of the monkey. Success rates in this task were at the 65-85% range. Cortical units were sorted in an automated manner every day with additional experimenter tuning. The number of very well-isolated single units ranged between 21-41. The remaining units consisted of 140-150 medium quality and multi-units, which the sorting software often split into more than one channel.

Most of the different algorithms that are compared here have free hyper-parameters that need to be set (such as a Gaussian kernel width for spike rates, the maximal informative time window of neural activities, $s$ and the $c$ trade-off parameter). We had a rough estimate for some of these from previous experiments using similar data (see [8]). To fine-tune these parameters, a brute-force grid search was performed on data from one of the 11 sessions in a (batch) 5-fold cross validation scheme. Those parameters were then kept fixed.

Earlier experiments showed the Gaussian kernel to be a good candidate for comparing neural spike rate vectors. It can also be calculated quickly, which is important for the BMI real-time constraint. We tested several variations of structured kernels on neuro-movement inputs. These variations consisted of all combinations of summing or multiplying Gaussian or linear kernels for the spike rates and movement states. Taking a sum of Gaussians or their product produced the best results (with no significant difference between these two options). We chose the sum (having the conceptual inference form: $\hat{\mathbf{y}}_t^i = \mathbf{W}_\psi \psi(\hat{\mathbf{y}}_{t-r:t-1}^i) + \mathbf{W}_\phi \phi(\mathbf{x}_{t-s:t}^i)$ where $\psi, \phi$ are the infinite feature vectors of the Gaussian kernel). The next best result was achieved by summing a Gaussian spike rate kernel and a linear movement kernel (which we will call lin-$\mathbf{y}$-KARMA). The sum of linear kernels produces ARMA (which we also tested). The test results that are presented in this study are only for the remaining 10 recording sessions. The main performance measure that we use here is the (Pearson) correlation coefficient (CC) between true and estimated values of positions (in each of the 3 movement dimensions). To gauge changes in prediction quality over time we use CC in a running window of sequences (window size is chosen so as to decrease the noise in the CC estimate). In other cases, the CC for a whole data-set is computed.

To illustrate KARMA's performance we provide a movie (see video 1 in supplementary material) showing the true hand position (in black) and KARMA's tracking estimate (in blue) during a continuous 150 second sequence of target-to target reach movements. This clip is of a model that was learned in an online manner using the previous (180) sequences, using a support vector cache of 3000 (as described in section 3). The initial position of the cursor is not given to the algorithm. Instead the average start positions in previous sequences is given as the starting point. The CC in the time window of the previous 40 sequences (0.9, 0.92 and 0.96 for the 3 dimensions) is given to provide a feeling of what such CC's look like. Similarly, Figure 1.B shows tracking and true position values for an 80 second segment towards the end of a different session.

KARMA achieves good performance and it does so with relatively small amounts of data. Figure 1.A shows tracking quality in terms of a running window of CC's over time. CC's for the first sequences are calculated on predictions made up to those times. While these CC's are more noisy it is clear that a useful model is reached within the first 3 minutes (CC's all higher than 0.6) and close to peak performance is already available within the first 10 minutes.

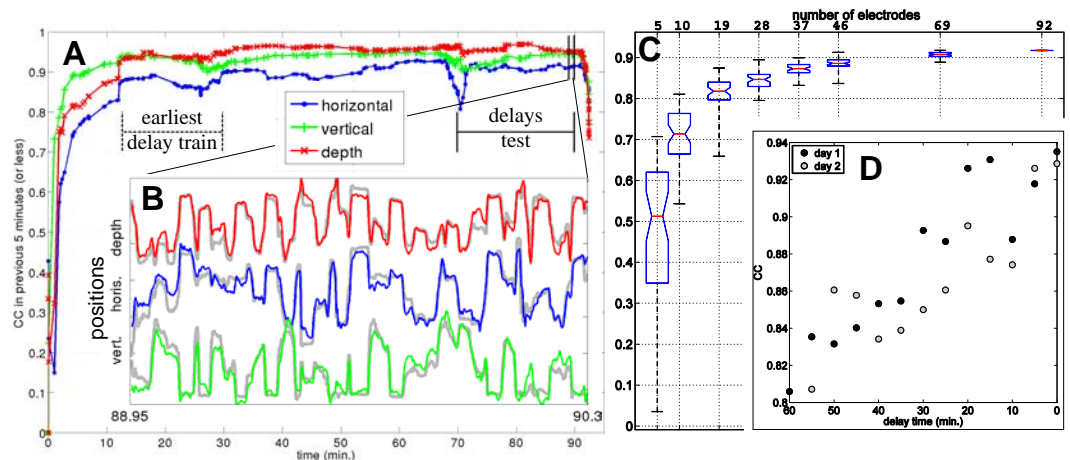

Figure 1: KARMA performance: (**A**) Correlation coefficients in a running window of 20 sequences (or less at the session start) for a whole 95 minute session (mean CC window size is 9.7 minutes). (**B**) True (gray) vs. tracked positions in an 80 second segment at minute 90 of the session. (**C**) Effect of loosing recording electrodes: tracking was performed over a full recording session using randomly chosen subsets of electrodes. For each selected number of electrodes (from the full 92 available down to 5 electrodes) 50 repetitions were run. CC's were calculated per run over the last two thirds of the session (to avoid effects of initial performance transients) and averaged over the 3 movement dimensions. Their distributions (over repetitions) are shown in terms of the median CC (red center line), quartile values (skirt containing 50% of the CC's) and extremal values (whiskers) for each number of electrodes. (**D**) Effect of delay time between training data and test data: For the session shown in (A), marked 'day 1' and for another session (marked 'day 2'), hand movement in 20 minute time windows towards the session ends were reconstructed in an online manner but instead of using the same sequences as training data (with a 1 step delay), earlier sequences were used. Figure (A) shows the time window that corresponded to opening a maximal time difference of 60 minutes between the last inferred sequence (at minute 90) and the last learned sequence (at minute 30). CC's for the test windows (averaged over movement dimensions) are shown as a function of delay time for the two days.

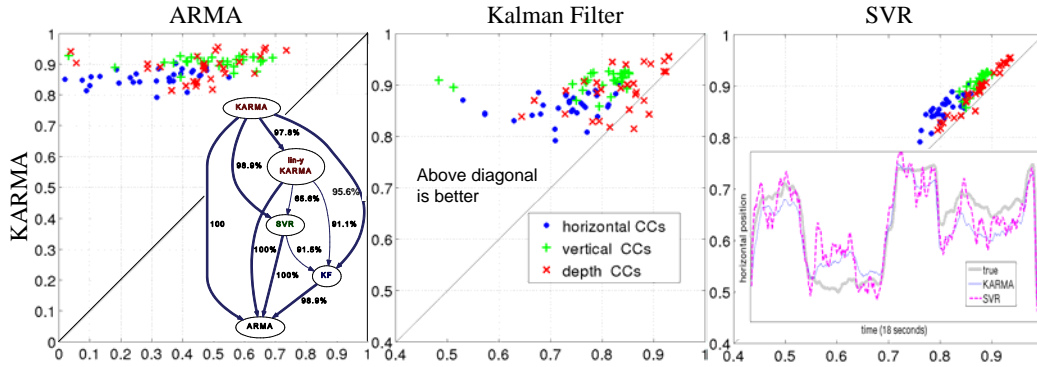

Figure 2: Algorithm comparisons: 10 hand-movement sessions were each divided into 3 equally long blocks of 25-35 minutes (the last few minutes were discarded since during this time the monkey often stopped paying attention to the task) to create 30 data-sets. The following algorithms were run on each data-set in an online manner: KARMA, lin-$\mathbf{y}$-KARMA, ARMA and SVR. All four were implemented as versions of the KARMA by varying its parameters. In all cases a support vector cache of 5000 was enforced as described in section 4. A Kalman Filter was also implemented so as to allow for a window of observations as input and a window of movement positions as the state (this version was previously shown to outperform the standard Kalman Filter which has $r = s = 1$). It was also learned in an online manner, replacing inverses with pseudo-inverses where necessary to avoid non-invertible matrices when data-sets are too small. Results are shown as scatter plots of CC values (30 data-sets and 3 movement dimensions produce 90 points per scatter plot). Each point compares KARMA to another algorithm in a specific data-set and movement dimension pair. Points above the diagonal mean a higher score for KARMA. The Graph on the left shows win-scores for each pair of algorithm. Win-score is defined at the percentage of times one algorithm got a higher CC than another. Edge direction points to the loser. The movement reconstruction on the right (horizontal position only) shows KARMA vs. SVR in a sample 18 second window.

Probably the highest source of variability in BMI performance across subjects is the location and number of electrodes in the brain. To test how performance with KARMA would degrade if electrodes were lost we simulated an electrode dropping experiment (see figure 1.C). Let's consider a CC of 0.7 as a minimal reasonable performance quality. Let's also assume that with 50 repetitions, minimal values roughly represent a worst case scenario in terms of mishaps that do not involve movement of the whole array. Then it seems that we can get by easily with only a third of the array (28 electrodes) operational. In terms of medians (average bad luck) we could do with less. Maximal values are relevant in case we need to choose the good electrodes. This may be relevant in situations involving implanted chips that extract and wirelessly transmit neural activity and may have constraints in terms of energy expenditure or bandwidth.

Most BMI experiments (e.g. [2, 13] with the exception of [1]) use fixed models that are learned once at the beginning of a session. Our version of KARMA is adaptive. In order to check the importance of adapting to changes in the recorded neural activity we ran an experiment in which variable delays were opened between the inference times and the latest available data for learning. i.e. after inference of sequence $\mathbf{y}^i$ from $\mathbf{x}^i$, sequence pair $\{\mathbf{x}^{i-k}, \mathbf{y}^{i-k}\}$ where $k > 0$ was first made available. Figure 1.D shows a degradation in performance during the test periods as the delay grows for two recording sessions. This suggests that adaptability of the algorithm is important for keeping high performance levels. There are two possible reasons for the observed degradation. One is changes in the neural activity within the brain. The other is changes in the process that extracts the neural activity in the BMI (e.g. electrode movements). Differentiating between the two options is a subject of future work. In the BMI setting, feedback is involved. The subject might be able to effortlessly modulate his neural activity and keep it in good fit with the algorithm. In section 5 we address this issue by running BMI sessions in which the model was frozen.

Comparison of KARMA to other methods is shown in figure 2. It is clear that KARMA performs much better than ARMA and the Kalman Filter, suggesting that a nonlinear interpretation of neural activity is helpful. While KARMA is statistically significantly better than SVR, the differences in CC values are not very big (note the scaling difference of the scatter plots). Looking at the movement reconstruction comparison it seems that SVR has a good average estimate of the current position,

however, missing a movement model (SVR has the form $\hat{\mathbf{y}}_t^i = \mathbf{W}_\phi \phi(\mathbf{x}_{t-s:t}^i)$ ) it fluctuates rapidly around the true value. This fluctuation may not be very apparent in the CC values however it would make a BMI much more difficult to control. Lin-$\mathbf{y}$-KARMA uses a linear movement model (and has the form: $\hat{\mathbf{y}}_t^i = \mathbf{A}\hat{\mathbf{y}}_{t-r:t-1}^i + \mathbf{W}_\phi \phi(\mathbf{x}_{t-s:t}^i)$). Its performance is inferior to full KARMA. Having a nonlinear movement model means that different areas of the state-space get treated in locally relevant fashion. This might explain why full KARMA outperforms. Note that the difference between lin-$\mathbf{y}$-KARMA and SVR are not very significant (win-score of only 65.6%). Comparison to the Population Vector algorithm was also done however the PVA achieved especially bad results for our long sequences since it accumulates errors without any decay (this is less of a problem in BMI since the subject can correct accumulated errors ). We therefore omit showing them here.

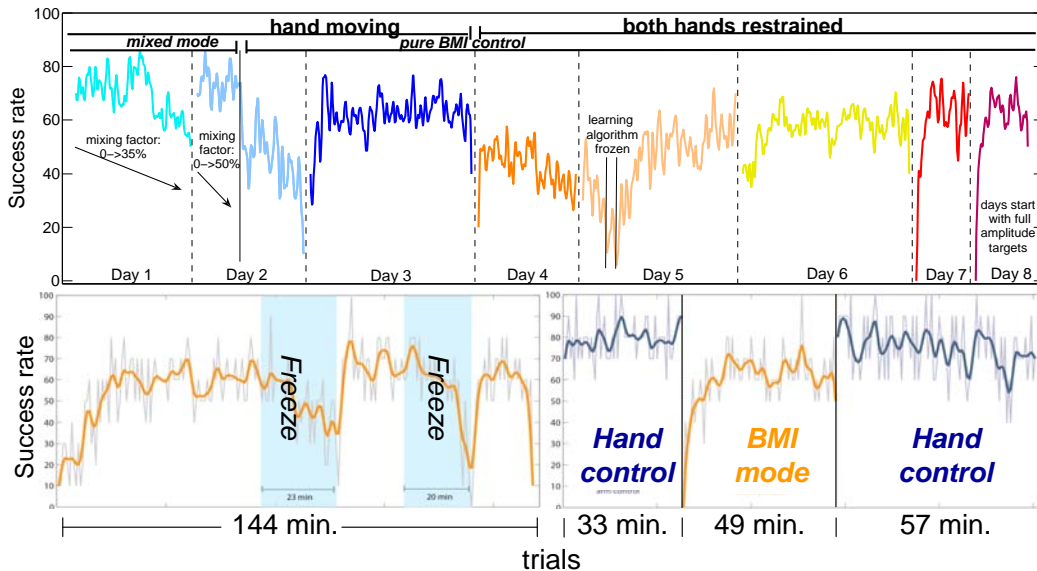

Figure 3: All graphs show success rates. The light, noisy plots are success rates in consecutive bins of 10 trials while the smoother plots are the result of running a smoothing filter on the noisy plots. Mixing mode was used on day 1 and part of day 2. Afterwards we switched to full BMI (mixing factor 100%). Hands were allowed to move freely until day 4. On day 4 both hands were comfortably restrained for the first time and though performance levels dropped, the monkey did not attempt to move its hands. On day 5 an attempt to freeze the model was made. When performance dropped and the monkey became agitated we restarted the BMI from scratch and performance improved rapidly. Day 6 consists of full BMI but with the targets not as far apart as with hand-control. This makes the task a bit easier and allowed for higher success rates. On days 7 and 8 a BMI block was interleaved with hand control blocks. Only the BMI blocks are shown in the top graph. The full three blocks of day 8 are shown in the bottom right graph. Bottom left graph shows a recording session during which the model was frozen.

## 5 BMI experiment

The BMI experiment was conducted after approximately four months of neural recordings during which the monkey learned and performed the hand-control task. Figure 3 shows a trace of the first days of the BMI task. A movie showing footage from these days is in the supplementary material (video 2). To make the transition to BMI smooth, the first 1.5 days consisted of work in a mixed mode, during which control of the cursor was a linear combination of the hand position and KARMA's predictions. We did this in order to see if the monkey would accept a noisier control scheme than it was used to. During the next 1.5 days the cursor was fully controlled by KARMA, but the monkey kept moving as if it was doing hand-control. i.e. it made movements and corrections with its hands. On day 4 the monkey's free hand was comfortably restrained. Despite our concerns that the monkey would stop performing it seemed impervious to this change, except that control over the cursor became more difficult. On days 5 and 6 we made some tweaks to the algorithm (tried to freeze learning and change the way in which correct movement trajectories are generated) and the

task (tried to decrease target size and the distance between targets) which had some effects of task difficulty and on success rates. On days 8 and 9 we interleaved a BMI block with hand-control blocks. We saw that performance is better in hand-control than in BMI but not drastically so. In the following sessions we discontinued all tweaking with the algorithm and we've seen some steady improvement in performance.

We repeated the freezing of model learning on two days (one of these session appears in figure 3). In all cases where we froze the model, we noticed that the monkey starts experiencing difficulty in controlling the cursor after a period of 10-15 minutes and stopped working completely when this happened. As stated earlier, in most BMI experiments the subjects interact with fixed models. One possible explanation for the deterioration is that because our monkey is trained in using an adaptive model it does not have experience in conforming to such a model's constraints. Instead, the opposite burden (that of following a drifting source of neural activity) falls on the algorithm's shoulders.

As mentioned in section 2, in BMI mode no hand movements are performed and therefore model learning is based on our guess of what is the desired cursor trajectory (the monkey's emphintended cursor movement). We chose to design the desired trajectory as a time varying linear combination of the cursor trajectory that the monkey saw and the target location: $\mathbf{y}_t^i = \left(1 - \frac{t}{t_{f_i}}\right)\hat{\mathbf{y}}_t^i + \frac{t}{t_{f_i}}\tilde{\mathbf{y}}^i$ where $\tilde{\mathbf{y}}^i$ is the target location on trial $i$. Note that this trajectory starts at the observed cursor location at the trial start and ends at the target location (regardless of where the cursor actually was at the end of the trial).

# 6    Summary

This study was motivated by the view that lifting overly simplifying assumptions and integrating domain knowledge into machine learning algorithms can make significant improvements to BMI technology. In turn, this technology can be used as a testbed for improved modeling of the interaction between the brain and environment, especially in visuomotor control. We showed in open-loop hand tracking experiments that the incorporation of a nonlinear movement model, interpreting the neural activity as a whole (rather than as the sum of contributions made by single neurons) and allowing the model to adapt to changes, results in better predictions. The comparison was done against similar models that lack one or more of these properties. Finally, we showed that this model can be used successfully in a real-time BMI setting, and that the added mathematical 'complications' result in a very intuitive and high quality interface.

# References

[1] D. M. Taylor, S. I. Helms, S.I. Tillery, and A.B. Schwartz. Direct cortical control of 3d neuroprosthetic devices. *Science*, 296(7):1829–1832, 2002.

[2] M.Velliste, S.Perel, M.C.Spalding, A.S. Whitford, and A. B. Schwartz. Cortical control of a prosthetic arm for self-feeding. *Nature (online)*, May 2008.

[3] S. Kakei, D. S. Hoffman, and P. L. Strick. Muscle and movement representations in the primary motor cortex. *Science*, 285:2136–2139, 1999.

[4] E. Vaadia, I. Haalman, M. Abeles, H. Bergman, Y. Prut, H. Slovin, and A. Aertsen. Dynamics of neuronal interactions in monkey cortex in relation to behavioral events. *Nature*, 373:515–518, Febuary 1995.

[5] W. Wu, Y. Gao, E. Bienenstock, J.P. Donoghue, and M.J. Black. Bayesian population coding of motor cortical activity using a kalman filter. *Neural Computation*, 18:80–118, 2005.

[6] A. E. Brockwell, A. L. Rojas, and R. E. Kass. Recursive Bayesian Decoding of Motor Cortical Signals by Particle Filtering. *J Neurophysiol*, 91(4):1899–1907, 2004.

[7] L. Shpigelman, Y. Singer, R. Paz, and E. Vaadia. Spikernels: Predicting hand movements by embedding population spike rates in inner-product spaces. *Neural Computation*, 17(3), 2005.

[8] L. Shpigelman, K.Crammer, R. Paz, E.Vaadia, and Y.Singer. A temporal kernel-based model for tracking hand movements from neural activities. In *NIPS*. 2005.

[9] P.M.L. Drezet and R.F. Harrison. Support vector machines for system identification. In *UKACC*, volume 1, pages 688–692, Sep 1998.

[10] M. Martínez-Ramón, J. Luis Rojo-Álvarez, G. Camps-Valls, J. Muñoz-Marí, A. Navia-Vázquez, E. Soria-Olivas, and A. R. Figueiras-Vidal. Support vector machines for nonlinear kernel ARMA system identification. *IEEE Trans. Neural Net.*, 17(6):1617–1622, 2006.

[11] G. E. P. Box and G. M. Jenkins. *Time Series Analysis: Forecasting and Control*. Prentice Hall PTR, Upper Saddle River, NJ, USA, 1994.

[12] B. Schoelkopf and A. J. Smola. *Learning with Kernels*. The MIT Press, Cambridge, MA, 2002.

[13] L.R.Hochberg, M.D. Serruya, G. M. Friehsand J.A. Mukand, M.Saleh, A. H. Caplan, A. Branner, D. Chen, R. D. Penn, and J. P. Donoghue. Neuronal ensemble control of prosthetic devices by a human with tetraplegia. *Nature*, 442:164–171, 2006.

